# Spiking Inputs to a Winner-take-all Network

**Matthias Oster and Shih-Chii Liu**
Institute of Neuroinformatics
University of Zurich and ETH Zurich
Winterthurerstrasse 190
CH-8057 Zurich, Switzerland
{mao,shih}@ini.phys.ethz.ch

## Abstract

Recurrent networks that perform a winner-take-all computation have been studied extensively. Although some of these studies include spiking networks, they consider only analog input rates. We present results of this winner-take-all computation on a network of integrate-and-fire neurons which receives spike trains as inputs. We show how we can configure the connectivity in the network so that the winner is selected after a pre-determined number of input spikes. We discuss spiking inputs with both regular frequencies and Poisson-distributed rates. The robustness of the computation was tested by implementing the winner-take-all network on an analog VLSI array of 64 integrate-and-fire neurons which have an innate variance in their operating parameters.

## 1 Introduction

Recurrent networks that perform a winner-take-all computation are of great interest because of the computational power they offer. They have been used in modelling attention and recognition processes in cortex [Itti et al., 1998, Lee et al., 1999] and are thought to be a basic building block of the cortical microcircuit [Douglas and Martin, 2004]. Descriptions of theoretical spike-based models [Jin and Seung, 2002] and analog VLSI (aVLSI) implementations of both spike and non-spike models [Lazzaro et al., 1989, Indiveri, 2000, Hahnloser et al., 2000] can be found in the literature. Although the competition mechanism in these models uses spike signals, they usually consider the external input to the network to be either an analog input current or an analog value that represents the spike rate.

We describe the operation and connectivity of a winner-take-all network that receives input spikes. We consider the case of the hard winner-take-all mode, where only the winning neuron is active and all other neurons are suppressed. We discuss a scheme for setting the excitatory and inhibitory weights of the network so that the winner which receives input with the shortest inter-spike interval is selected after a pre-determined number of input spikes. The winner can be selected with as few as two input spikes, making the selection process fast [Jin and Seung, 2002].

We tested this computation on an aVLSI chip with 64 integrate-and-fire neurons and various dynamic excitatory and inhibitory synapses. The distribution of mismatch (or variance) in the operating parameters of the neurons and synapses has been reduced using a spike coding

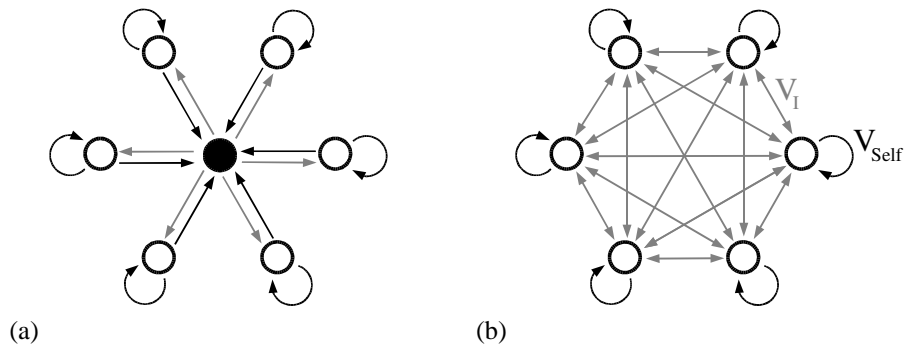

(a)                                                     (b)

Figure 1: Connectivity of the winner-take-all network: (a) in biological networks, inhibition is mediated by populations of global inhibitory interneurons (filled circle). To perform a winner-take-all operation, they are driven by excitatory neurons (unfilled circles) and in return, they inhibit all excitatory neurons (black arrows: excitatory connections; dark arrows: inhibitory). (b) Network model in which the global inhibitory interneuron is replaced by full inhibitory connectivity of efficacy $V_I$. Self excitation of synaptic efficacy $V_{self}$ stabilizes the selection of the winning neuron.

mismatch compensation procedure described in [Oster and Liu, 2004]. The results shown in Section 3 of this paper were obtained with a network that has been calibrated so that the neurons have about 10% variance in their firing rates in response to a common input current.

## 1.1   Connectivity

We assume a network of integrate-and-fire neurons that receive external excitatory or inhibitory spiking input. In biological networks, inhibition between these array neurons is mediated by populations of global inhibitory interneurons (Fig. 1a). They are driven by the excitatory neurons and inhibit them in return. In our model, we assume the forward connections between the excitatory and the inhibitory neurons to be strong, so that each spike of an excitatory neuron triggers a spike in the global inhibitory neurons. The strength of the total inhibition between the array neurons is adjusted by tuning the backward connections from the global inhibitory neurons to the array neurons. This configuration allows the fastest spreading of inhibition through the network and is consistent with findings that inhibitory interneurons tend to fire at high frequencies.

With this configuration, we can simplify the network by replacing the global inhibitory interneurons with full inhibitory connectivity between the array neurons (Fig. 1b). In addition, each neuron has a self-excitatory connection that facilitates the selection of this neuron as winner for repeated input.

## 2   Network Connectivity Constraints for a Winner-Take-All Mode

We first discuss the conditions for the connectivity under which the network operates in a hard winner-take-all mode. For this analysis, we assume that the neurons receive spike trains of regular frequency. We also assume the neurons to be non-leaky.

The membrane potentials $V_i$, $i = 1 \ldots N$ then satisfy the equation of a non-leaky integrate-

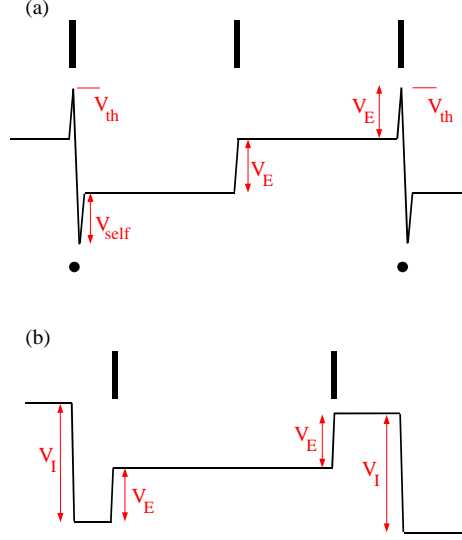

Figure 2: Membrane potential of the winning neuron $k$ (a) and another neuron in the array (b). Black bars show the times of input spikes. Traces show the changes in the membrane membrane potential caused by the various synaptic inputs. Black dots show the times of output spikes of neuron $k$.

and-fire neuron model with non-conductance-based synapses:

$$\frac{\mathrm{d}V_i}{\mathrm{d}t} = V_E \sum_n \delta(t - t_i^{(n)}) - V_I \sum_{\substack{j=1 \\ j \neq i}}^{N} \sum_m \delta(t - s_j^{(m)}) \tag{1}$$

The membrane resting potential is set to $0$. Each neuron receives external excitatory input and inhibitory connections from all other neurons. All inputs to a neuron are spikes and its output is also transmitted as spikes to other neurons. We neglect the dynamics of the synaptic currents and the delay in the transmission of the spikes. Each input spike causes a fixed discontinuous jump in the membrane potential ($V_E$ for the excitatory synapse and $V_I$ for the inhibitory). Each neuron $i$ spikes when $V_i \geq V_{th}$ and is reset to $V_i = 0$. Immediately afterwards, it receives a self-excitation of weight $V_{self}$. All potentials satisfy $0 \leq V_i \leq V_{th}$, that is, an inhibitory spike can not drive the membrane potential below ground. All neurons $i \in 1 \ldots N, i \neq k$ receive excitatory input spike trains of constant frequency $r_i$. Neuron $k$ receives the highest input frequency ($r_k > r_i \; \forall \; i \neq k$).

As soon as neuron $k$ spikes once, it has won the computation. Depending on the initial conditions, other neurons can at most have transient spikes before the first spike of neuron $k$. For this hard winner-take-all mode, the network has to fulfill the following constraints (Fig. 2):

(a) Neuron $k$ (the winning neuron) spikes after receiving $n_k = n$ input spikes that cause its membrane potential to exceed threshold. After every spike, the neuron is reset to $V_{self}$:

$$V_{self} + n_k V_E \geq V_{th} \tag{2}$$

(b) As soon as neuron $k$ spikes once, no other neuron $i \neq k$ can spike because it receives an inhibitory spike from neuron $k$. Another neuron can receive up to $n$ spikes even if its input spike frequency is lower than that of neuron $k$ because the neuron is reset to $V_{self}$

after a spike, as illustrated in Figure 2. The resulting membrane voltage has to be smaller than before:

$$n_i \cdot V_E \leq n_k \cdot V_E \leq V_I \tag{3}$$

(c) If a neuron $j$ other than neuron $k$ spikes in the beginning, there will be some time in the future when neuron $k$ spikes and becomes the winning neuron. From then on, the conditions (a) and (b) hold, so a neuron $j \neq k$ can at most have a few transient spikes.

Let us assume that neurons $j$ and $k$ spike with almost the same frequency (but $r_k > r_j$). For the inter-spike intervals $\Delta_i = 1/r_i$ this means $\Delta_j > \Delta_k$. Since the spike trains are not synchronized, an input spike to neuron $k$ has a changing phase offset $\phi$ from an input spike of neuron $j$. At every output spike of neuron $j$, this phase decreases by $\Delta\phi = n_k(\Delta_j - \Delta_k)$ until $\phi < n_k(\Delta_j - \Delta_k)$. When this happens, neuron $k$ receives $(n_k + 1)$ input spikes before neuron $j$ spikes again and crosses threshold:

$$(n_k + 1) \cdot V_E \geq V_{th} \tag{4}$$

We can choose $V_{self} = V_E$ and $V_I = V_{th}$ to fulfill the inequalities (2)-(4). $V_E$ is adjusted to achieve the desired $n_k$.

Case (c) happens only under certain initial conditions, for example when $V_k \ll V_j$ or when neuron $j$ initially received a spike train of higher frequency than neuron $k$. A leaky integrate-and-fire model will ensure that all membrane potentials are discharged ($V_i = 0$) at the onset of a stimulus. The network will then select the winning neuron after receiving a pre-determined number of input spikes and this winner will have the first output spike.

## 2.1 Poisson-Distributed Inputs

In the case of Poisson-distributed spiking inputs, there is a probability associated with the correct winner being selected. This probability depends on the Poisson rate $\nu$ and the number of spikes needed for the neuron to reach threshold $n$. The probability that $m$ input spikes arrive at a neuron in the period $T$ is given by the Poisson distribution

$$\mathrm{P}(m, \nu T) = e^{-\nu T} \frac{(\nu T)^m}{m!} \tag{5}$$

We assume that all neurons $i$ receive an input rate $\nu_i$, except the winning neuron which receives a higher rate $\nu_k$. All neurons are completely discharged at $t = 0$.

The network will make a correct decision at time $T$, if the winner crosses threshold exactly then with its $n$th input spike, while all other neuron received less than $n$ spikes until then.

The winner receives the $n$th input spike at $T$, if it received $n-1$ input spikes in $[0; T[$ and one at time $T$. This results in the probability density function

$$p_k(T) = \nu_k \mathrm{P}(n-1, \nu_k T) \tag{6}$$

The probability that the other $\mathrm{N}-1$ neurons receive less or equal than $n-1$ spikes in $[0; T[$ is

$$P_0(T) = \prod_{\substack{i=1 \\ i \neq k}}^{\mathrm{N}} \left( \sum_{j=0}^{n-1} \mathrm{P}(j, \nu_i T) \right) \tag{7}$$

For a correct decision, the output spike of the winner can happen at any time $T > 0$, so we integrate over all times $T$:

$$P = \int_0^\infty p_k(T) \cdot P_0(T) \, \mathrm{d}T = \int_0^\infty \nu_k \mathrm{P}(n-1, \nu_k T) \cdot \prod_{\substack{j=1 \\ i \neq k}}^{\mathrm{N}} \left( \sum_{i=0}^{n-1} \mathrm{P}(j, \nu_i T) \right) \mathrm{d}T \tag{8}$$

We did not find a closed solution for this integral, but we can discuss its properties $n$ is varied by changing the synaptic efficacies. For $n = 1$ every input spike elicits an output spike. The probability of a having an output spike from neuron $k$ is then directly dependent on the input rates, since no computation in the network takes place. For $n \rightarrow \infty$, the integration times to determine the rates of the Poisson-distributed input spike trains are large, and the neurons perform a good estimation of the input rate. The network can then discriminate small changes in the input frequencies. This gain in precision leads a slow response time of the network, since a large number of input spike is integrated before an output spike of the network.

The winner-take-all architecture can also be used with a latency spike code. In this case, the delay of the input spikes after a global reset determines the strength of the signal. The winner is selected after the first input spike to the network ($n_k = 1$). If all neurons are discharged at the onset of the stimulus, the network does not require the global reset. In general, the computation is finished at a time $n_k \cdot \Delta_k$ after the stimulus onset.

## 3   Results

We implemented this architecture on a chip with 64 integrate-and-fire neurons implemented in analog VLSI technology. These neurons follow the model equation 1, except that they also show a small linear leakage. Spikes from the neurons are communicated off-chip using an asynchronous event representation transmission protocol (AER). When a neuron spikes, the chip outputs the address of this neuron (or spike) onto a common digital bus (see Figure 3). An external spike interface module (consisting of a custom computer board that can be programmed through the PCI bus) receives the incoming spikes from the chip, and retransmits spikes back to the chip using information stored in a routing table. This module can also monitor spike trains from the chip and send spikes from a stored list. Through this module and the AER protocol, we implement the connectivity needed for the winner-take-all network in Figure 1. All components have been used and described in previous work [Boahen, 2000, Liu et al., 2001].

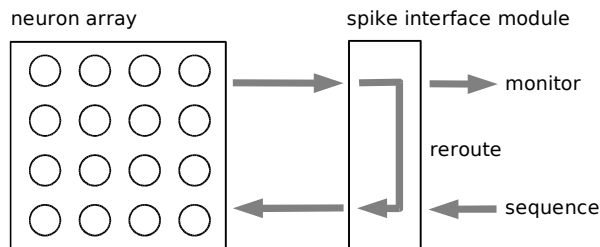

Figure 3: The connections are implemented by transmitting spikes over a common bus (grey arrows). Spikes from aVLSI neurons in the network are recorded by the digital interface and can be monitored and rerouted to any neuron in the array. Additionally, externally generated spike trains can be transmitted to the array through the sequencer.

We configure this network according to the constraints which are described above. Figure 4 illustrates the network behaviour with a spike raster plot. At time $t = 0$, the neurons receive inputs with the same regular firing frequency of 100Hz except for one neuron which received a higher input frequency of 120Hz. The synaptic efficacies were tuned so that threshold is reached with 6 input spikes, after which the network does select the neuron with the strongest input as the winner.

We characterized the discrimination capability of the winner-take-all implementation by

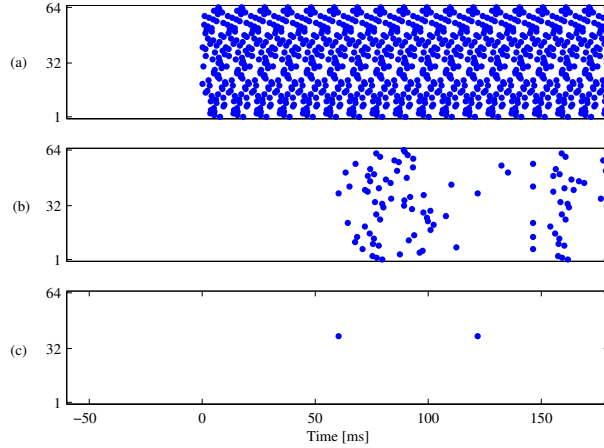

Figure 4: Example raster plot of the spike trains to and from the neurons: (a) Input: starting from 0 ms, the neurons are stimulated with spike trains of a regular frequency of 100Hz, but randomized phase. Neuron number 42 receives an input spike train with an increased frequency of 120Hz. (b) Output without WTA connectivity: after an adjustable number of input spikes, the neurons start to fire with a regular output frequency. The output frequencies of the neurons are slightly different due to mismatch in the synaptic efficacies. Neuron 42 has the highest output frequency since it receives the strongest input. (c) Output with WTA connectivity: only neuron 42 with the strongest input fires, all other neurons are suppressed.

measuring to which minimal frequency, compared to the other input, the input rate to this neuron has to be raised to select it as the winner. The neuron being tested receives an input of regular frequency of $f \cdot 100$Hz, while all other neuron receive 100Hz. The histogram of the minimum factors $f$ for all neurons is shown in Figure 5. On average, the network can discriminate a difference in the input frequency of 10%. This value is identical with the variation in the synaptic efficacies of the neurons, which had been compensated to a mismatch of 10%. We can therefore conclude that the implemented winner-take-all network functions according to the above discussion of the constraints. Since only the timing information of the spike trains is used, the results can be extended to a wide range of input frequencies different from 100Hz.

To test the performance of the network with Poisson inputs, we stimulated all neurons with Poisson-distibuted spike rates of rate $\nu$, except neuron $k$ which received the rate $\nu_k = f\nu$. Eqn. 8 then simplifies to

$$P = \int_0^\infty f\nu\, \mathrm{P}(n-1, f\nu\, T) \cdot \left( \sum_{i=0}^{n-1} P(i, \nu T) \right)^{N-1} \mathrm{d}T \qquad (9)$$

We show measured data and theoretical predictions for a winner-take-all network of 2 and 8 neurons (Fig. 6). Obviously, the discrimation performance of the network is substantially limited by the Poisson nature of the spike trains compared to spike trains of regular frequency.

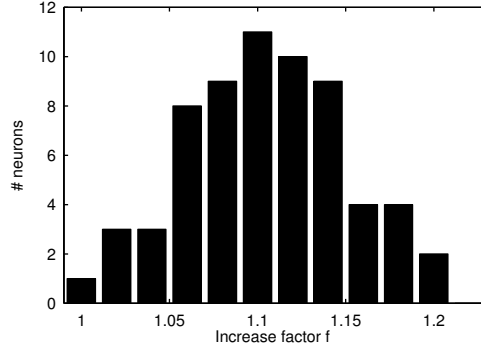

Figure 5: Discrimination capability of the winner-take-all network: X-axis: factor $f$ to which the input frequency of a neuron has to be increased, compared to the input rate of the other neurons, in order for that neuron to be selected as the winner. Y-axis: histogram of all 64 neurons.

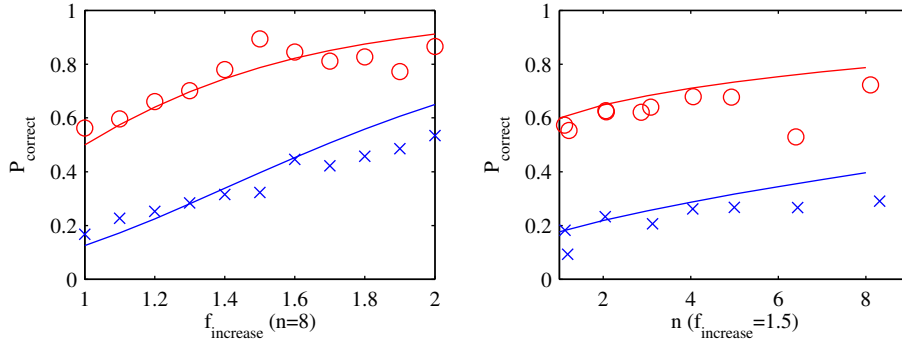

Figure 6: Probability of a correct decision of the winner-take-all network, versus difference in frequencies (left), and number of input spikes $n$ for a neuron to reach threshold (right). The measured data (crosses/circles) is shown with the prediction of the model (continuous lines), for a winner-take-all network of 2 neurons (red,circles) and 8 neurons (blue, crosses).

## 4 Conclusion

We analysed the performance and behavior of a winner-take-all spiking network that receives input spike trains. The neuron that receives spikes with the highest rate is selected as the winner after a pre-determined number of input spikes. Assuming a non-leaky integrate-and-fire model neuron with constant synaptic weights, we derived constraints for the strength of the inhibitory connections and the self-excitatory connection of the neuron. A large inhibitory synaptic weight is in agreement with previous analysis for analog inputs [Jin and Seung, 2002]. The ability of a single spike from the inhibitory neuron to inhibit all neurons removes constraints on the matching of the time constants and efficacy of the connections from the excitatory neurons to the inhibitory neuron and vice versa. This feature makes the computation tolerant to variance in the synaptic parameters as demonstrated by the results of our experiment.

We also studied whether the network is able to select the winner in the case of input spike trains which have a Poisson distribution. Because of the Poisson distributed inputs, the network does not always chose the right winner (that is, the neuron with the highest input

frequency) but there is a certain probability that the network does select the right winner. Results from the network show that the measured probabilities match that of the theoretical results. We are currently extending our analysis to a leaky integrate-and-fire neuron model and conductance-based synapses, which results in a more complex description of the network.

### Acknowledgments

This work was supported in part by the IST grant IST-2001-34124. We acknowledge Sebastian Seung for discussions on the winner-take-all mechanism.

## References

[Boahen, 2000] Boahen, K. A. (2000). Point-to-point connectivity between neuromorphic chips using address-events. *IEEE Transactions on Circuits & Systems II*, 47(5):416–434.

[Douglas and Martin, 2004] Douglas, R. and Martin, K. (2004). Cortical microcircuits. *Annual Review of Neuroscience*, 27(1f).

[Hahnloser et al., 2000] Hahnloser, R., Sarpeshkar, R., Mahowald, M. A., Douglas, R. J., and Seung, S. (2000). Digital selection and analogue amplification coexist in a cortex-inspired silicon circuit. *Nature*, 405:947–951.

[Indiveri, 2000] Indiveri, G. (2000). Modeling selective attention using a neuromorphic analog VLSI device. *Neural Computation*, 12(12):2857–2880.

[Itti et al., 1998] Itti, C., Niebur, E., and Koch, C. (1998). A model of saliency-based fast visual attention for rapid scene analysis. *IEEE Transactions on Pattern Analysis and Machine Intelligence*, 20(11):1254–1259.

[Jin and Seung, 2002] Jin, D. Z. and Seung, H. S. (2002). Fast computation with spikes in a recurrent neural network. *Physical Review E*, 65:051922.

[Lazzaro et al., 1989] Lazzaro, J., Ryckebusch, S., Mahowald, M. A., and Mead, C. A. (1989). Winner-take-all networks of O(n) complexity. In Touretzky, D., editor, *Advances in Neural Information Processing Systems*, volume 1, pages 703–711. Morgan Kaufmann, San Mateo, CA.

[Lee et al., 1999] Lee, D., Itti, C., Koch, C., and Braun, J. (1999). Attention activates winner-take-all competition among visual filters. *Nature Neuroscience*, 2:375–381.

[Liu et al., 2001] Liu, S.-C., Kramer, J., Indiveri, G., Delbrück, T., Burg, T., and Douglas, R. (2001). Orientation-selective aVLSI spiking neurons. *Neural Networks: Special Issue on Spiking Neurons in Neuroscience and Technology*, 14(6/7):629–643.

[Oster and Liu, 2004] Oster, M. and Liu, S.-C. (2004). A winner-take-all spiking network with spiking inputs. In *11th IEEE International Conference on Electronics, Circuits and Systems*. ICECS '04: Tel Aviv, Israel, 13–15 December.
